# Euclidean Embedding of Co-occurrence Data

**Amir Globerson**[1] **Gal Chechik**[2] **Fernando Pereira**[3] **Naftali Tishby**[1]
[1] School of computer Science and Engineering,
Interdisciplinary Center for Neural Computation
The Hebrew University Jerusalem, 91904, Israel
[2] Computer Science Department, Stanford University, Stanford, CA 94305, USA
[3] Department of Computer and Information Science,
University of Pennsylvania, Philadelphia, PA 19104, USA

## Abstract

Embedding algorithms search for low dimensional structure in complex data, but most algorithms only handle objects of a single type for which pairwise distances are specified. This paper describes a method for embedding objects of different types, such as images and text, into a single common Euclidean space based on their co-occurrence statistics. The joint distributions are modeled as exponentials of Euclidean distances in the low-dimensional embedding space, which links the problem to convex optimization over positive semidefinite matrices. The local structure of our embedding corresponds to the statistical correlations via random walks in the Euclidean space. We quantify the performance of our method on two text datasets, and show that it consistently and significantly outperforms standard methods of statistical correspondence modeling, such as multidimensional scaling and correspondence analysis.

## 1   Introduction

Embeddings of objects in a low-dimensional space are an important tool in unsupervised learning and in preprocessing data for supervised learning algorithms. They are especially valuable for exploratory data analysis and visualization by providing easily interpretable representations of the relationships among objects. Most current embedding techniques build low dimensional mappings that preserve certain relationships among objects and differ in the relationships they choose to preserve, which range from pairwise distances in multidimensional scaling (MDS) [4] to neighborhood structure in locally linear embedding [12]. All these methods operate on objects of a single type endowed with a measure of similarity or dissimilarity.

However, real-world data often involve objects of several very different types without a natural measure of similarity. For example, typical web pages or scientific papers contain varied data types such as text, diagrams, images, and equations. A measure of similarity between words and pictures is difficult to define objectively. Defining a useful measure of similarity is even difficult for some homogeneous data types, such as pictures or sounds, where the physical properties (pitch and frequency in sounds, color and luminosity distribution in images) do not directly reflect the semantic properties we are interested in.

The current paper addresses this problem by creating embeddings from statistical associations. The idea is to find a Euclidean embedding in low dimension that represents the empirical co-occurrence statistics of two variables. We focus on modeling the conditional probability of one variable given the other, since in the data we analyze (documents and words, authors and terms) there is a clear asymmetry which suggests a conditional model. Joint models based on similar principles can be devised in a similar fashion, and may be more appropriate for symmetric data. We name our method CODE for *Co-Occurrence Data Embedding*.

Our cognitive notions are often built through statistical associations between different information sources. Here we assume that those associations can be represented in a low-dimensional space. For example, pictures which frequently appear with a given text are expected to have some common, locally low-dimensional characteristic that allows them to be mapped to adjacent points. We can thus rely on co-occurrences to embed different entity types, such as words and pictures, genes and expression arrays, into the same subspace. Once this embedding is achieved it also naturally defines a measure of similarity between entities of the same kind (such as images), induced by their other corresponding modality (such as text), providing a meaningful similarity measure between images.

Embedding of heterogeneous objects is performed in statistics using *correspondence analysis* (CA), a variant of *canonical correlation analysis* for count data [8]. These are related to Euclidean distances when the embeddings are constrained to be normalized. However, as we show below, removing this constraint has great benefits for real data. Statistical embedding of same-type objects was recently studied by Hinton and Roweis [9]. Their approach is similar to ours in that it assumes that distances induce probabilistic relations between objects. However, we do not assume that distances are given in advance, but instead we derive them from the empirical co-occurrence data. The Parametric Embedding method [11], which also appears in the current proceedings, is formally similar to our method but is used in the setting of supervised classification.

## 2   Problem Formulation

Let $X$ and $Y$ be two categorical variables with an empirical distribution $\bar{p}(x,y)$. No additional assumptions are made on the values of $X$ and $Y$ or their relationships. We wish to model the statistical dependence between $X$ and $Y$ through an intermediate Euclidean space $\mathbb{R}^d$ and mappings $\vec{\phi}: X \to \mathbb{R}^d$ and $\vec{\psi}: Y \to \mathbb{R}^d$. These mappings should reflect the dependence between $X$ and $Y$ in the sense that the distance between each $\vec{\phi}(x)$ and $\vec{\psi}(y)$ determines their co-occurrence statistics.

We focus in this manuscript on modeling the conditional distribution $p(y|x)$[1], and define a model which relates conditional probabilities to distances by

$$p(y|x) = \frac{\bar{p}(y)}{Z(x)} e^{-d_{x,y}^2} \quad \forall x \in X, \forall y \in Y \tag{1}$$

where $d_{x,y}^2 \equiv |\vec{\phi}(x) - \vec{\psi}(y)|^2 = \sum_{k=1}^{d}(\phi_k(x) - \psi_k(y))^2$ is the Euclidean distance between $\vec{\phi}(x)$ and $\vec{\psi}(y)$ and $Z(x)$ is the partition function for each value of $x$. This partition function equals $Z(x) = \sum_y \bar{p}(y) e^{-d_{x,y}^2}$ and is thus the empirical mean of the exponentiated distances from $x$ (therefore $Z(x) \leq 1$).

This model directly relates the ratio $\frac{p(y|x)}{\bar{p}(y)}$ to the distance between the embedded $x$ and $y$. The ratio decays exponentially with the distance, thus for any $x$, a closer $y$ will have

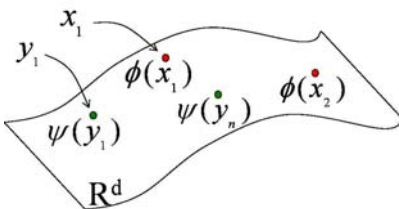

Figure 1: Embedding of $X, Y$ into the same $d$-dimensional space.

a higher interaction ratio. As a result of the fast decay, the closest objects dominate the distribution. The model of Eq. 1 can also be described as the result of a random walk in the low-dimensional space illustrated in Figure 1. When $y$ has a uniform marginal, the probability $p(y|x)$ corresponds to a random walk from $x$ to $y$, with transition probability inversely related to distance.

We now turn to the task of learning $\vec{\phi}, \vec{\psi}$ from an empirical distribution $\bar{p}(x, y)$. It is natural in this case to maximize the likelihood (up to constants depending on $\bar{p}(y)$)

$$\max_{\vec{\phi}, \vec{\psi}} l(\vec{\phi}, \vec{\psi}) = -\sum_{x,y} \bar{p}(x, y) d^2_{x,y} - \sum_x \bar{p}(x) \log Z(x) , \qquad (2)$$

where $\bar{p}(x, y)$ denotes the empirical distribution over $X, Y$. As in other cases, maximizing the likelihood is also equivalent to minimizing the $D_{KL}$ between the empirical and the model's distributions. The likelihood is composed of two terms. The first is (minus) the mean distance between $x$ and $y$. This will be maximized when all distances are zero. This trivial solution is avoided because of the *regularization* term $\sum_x \bar{p}(x) \log Z(x)$, which acts to increase distances between $x$ and $y$ points. The next section discusses the relation of this target function to that of Canonical Correlation Analysis [10].

To characterize the maxima of the likelihood we differentiate it with respect to the embeddings of individual objects $\vec{\phi}(x), \vec{\psi}(y)$, and obtain the following gradients

$$\frac{\partial l(\vec{\phi}, \vec{\psi})}{\partial \vec{\phi}(x)} = 2\bar{p}(x)\left(\langle \vec{\psi}(y)\rangle_{\bar{p}(y|x)} - \langle \vec{\psi}(y)\rangle_{p(y|x)}\right) \qquad (3)$$

$$\frac{\partial l(\vec{\phi}, \vec{\psi})}{\partial \vec{\psi}(y)} = 2p(y)\left(\vec{\psi}(y) - \langle \vec{\phi}(x)\rangle_{p(x|y)}\right) - 2\bar{p}(y)\left(\vec{\psi}(y) - \langle \vec{\phi}(x)\rangle_{\bar{p}(x|y)}\right) ,$$

where $p(y) = \sum_x p(y|x)\bar{p}(x)$.

Equating these gradients to zero, the $\vec{\phi}(x)$ gradient yields $\langle \vec{\psi}(y)\rangle_{p(y|x)} = \langle \vec{\psi}(y)\rangle_{\bar{p}(y|x)}$. This characterization is similar to the one seen in maximum entropy learning. Since $p(y|x)$ will have significant values for $Y$ values such that $\psi(y)$ is close to $\phi(x)$, this condition implies that the expected location of a neighbor of $\phi(x)$ is the same under the empirical and model distributions.

To find the optimal $\vec{\phi}, \vec{\psi}$ for a given embedding dimension $d$, we used a conjugate gradient ascent algorithm with random restarts. In section 4 we describe a different approach to this optimization problem.

## 3  Relation to Other Methods

Embedding the rows and columns of a contingency table into a low dimensional Euclidean space is related to statistical methods for the analysis of heterogeneous data. Fisher [6] described a method for mapping $X$ and $Y$ into $\phi(x), \psi(y)$ such that the correlation coefficient between $\phi(x), \psi(y)$ is maximized. His method is in fact the discrete analogue of the more widely known *Canonical correlation analysis* (CCA) [10]. Another closely related method is *Correspondence analysis* [8], which uses a different normalization scheme, and aims to model $\chi^2$ distances between rows and columns of $\bar{p}(x, y)$.

The goal of all the above methods is to maximize the correlation coefficient between the embeddings of $X$ and $Y$. We now discuss their relation to our *distance* based method. First, note that the correlation coefficient is invariant under affine transformations and we can thus focus on centered solutions with a unity covariance matrix ($\langle\phi(x)\rangle = 0$ and $COV(\phi(x)) = COV(\psi(y)) = I$) solutions. In this case, the correlation coefficient is given by the following expression (we focus on $d = 1$ for simplicity)

$$\rho(\phi(x), \psi(y)) = \sum_{x,y} \bar{p}(x,y)\phi(x)\psi(y) = -\frac{1}{2}\sum_{x,y} \bar{p}(x,y)d_{x,y}^2 + 1 \ . \tag{4}$$

Maximizing the correlation is therefore equivalent to minimizing the mean distance across all pairs. This clarifies the relation between CCA and our method: Both methods aim to minimize the average distance between $X$ and $Y$ embeddings. However, CCA forces both embeddings to be centered and with a unity covariance matrix, whereas our method introduces a global regularization term related to the partition function.

Our method is additionally related to exponential models of contingency tables, where the counts are approximated by a normalized exponent of a low rank matrix [7]. The current approach can be understood as a constrained version of these models where the expression in the exponent is constrained to have a geometric interpretation.

A well-known geometric oriented embedding method is multidimensional scaling (MDS) [4], whose standard version applies to same-type objects with predefined distances. MDS embedding of heterogeneous entities was studied in the context of modeling ranking data (see [4] section 7.3). These models, however, focus on specific properties of ordinal data and therefore result in optimization principles and algorithms different from our probabilistic interpretation.

## 4  Semidefinite Representation

The optimal embeddings $\vec{\phi}, \vec{\psi}$ may be found using unconstrained optimization techniques. However, the Euclidean distances used in the embedding space also allow us to reformulate the problem as constrained convex optimization over the cone of positive semidefinite (PSD) matrices [14].

We start by showing that for embeddings with dimension $d = |X| + |Y|$, maximizing (2) is equivalent to minimizing a certain convex non-linear function over PSD matrices. Consider the matrix $A$ whose columns are all the embedded vectors $\vec{\phi}$ and $\vec{\psi}$. The matrix $G \equiv A^T A$ is the Gram matrix of the dot products between embedding vectors. It is thus a symmetric PSD matrix of rank $\leq d$. The converse is also true: any PSD matrix of rank $\leq d$ can be factorized as $A^T A$, where $A$ is an embedding matrix of dimension $d$. The distance between two columns in $A$ is linearly related to the Gram matrix via $d_{ij}^2 = g_{ii} + g_{jj} - 2g_{ij}$.

Since the likelihood function depends only on the distances between points in $X$ and in $Y$,

we can write the optimization problem in (2) as

$$\min_{G} \sum_{x} \bar{p}(x) \log \sum_{y} \bar{p}(y) e^{-d_{xy}^2} + \sum_{x,y} \bar{p}(x,y) d_{xy}^2 \qquad (5)$$

$$\text{Subject to} \quad G \succeq 0, \quad \text{rank}(G) \leq d, \quad d_{xy}^2 = g_{xx} + g_{yy} - 2g_{xy}$$

where $g_{xy}$ denotes the element in $G$ corresponding to specific values of $x, y$.

Thus, our problem is equivalent to optimizing a nonlinear objective over the set of PSD matrices of a constrained rank. The minimized function is convex, since it is the sum of a linear function of $G$ and functions $\log \sum \exp$ of an affine expression in $G$, which are also convex (see Geometric Programming section in [2]). Moreover, when $G$ has full rank, the set of constraints is also convex. We conclude that when the embedding dimension is of size $d = |X| + |Y|$ the optimization problem of Eq. (5) is convex. Thus there are no local minima, and solutions can be found efficiently.

The PSD formulation allows us to add non-trivial constraints. Consider, for example, constraining the $p(y)$ marginal to its empirical values, i.e. $\sum_x p(y|x)\bar{p}(x) = \bar{p}(y)$. To introduce this as a convex constraint we take two steps. First, we note that we can relax the constraint that distributions normalize to one, and require that they normalize to less than one. This is achieved by replacing $\log Z(x)$ with a free variable $a(x)$ and writing the problem as follows (we omit the dependence of $d_{xy}^2$ on $G$ for brevity)

$$\min_{G} \sum_{x} \bar{p}(x) a(x) + \sum_{x,y} \bar{p}(x,y) d_{xy}^2 \qquad (6)$$

$$\text{Subject to} \quad G \succeq 0, \quad \text{rank(G)} \leq \text{d}, \quad \log \sum_{y} \bar{p}(y) e^{-d_{xy}^2 - a(x)} \leq 0 \quad \forall x$$

It can be shown that the optimum of 6 will be obtained for solutions normalized to one, and it thus coincides with the optimum of 5. The constraint $\sum_x p(y|x)\bar{p}(x) = \bar{p}(y)$ can now be relaxed to the *inequality* $\sum_x \bar{p}(y)\bar{p}(x) e^{-d_{xy}^2 - a(x)} \leq \bar{p}(y)$, which defines a convex set. Again, the optimum will be obtained when the constraint is satisfied with equality.

Embedding into a low dimension requires constraining the rank, but this is difficult since the problem is no longer convex in the general case. One approach to obtaining low rank solutions is to optimize over a full rank $G$ and then project it into a lower dimension via spectral decomposition as in [14] or classical MDS. However, in the current problem, this was found to be ineffective. Instead, we penalize high-rank solutions by adding the trace of $G$ [5] weighted by a positive factor, $\lambda$, to the objective function in (5). Small values of $\text{Tr}(G)$ are expected to correspond to sparse eigenvalue sets and thus penalize high rank solutions. This approach was tested on subsets of the databases described in section 5 and yielded similar results to those of the gradient based algorithm. We believe that PSD algorithms may turn out to be more efficient in cases where relatively high dimensional embeddings are sought. Furthermore, under the PSD formulation it is easy to introduce additional constraints, for example on distances between subsets of points (as in [14]), and on marginals of the distribution.

## 5 Applications

We tested our approach on a variety of applications. Here we present embedding of words and documents and authors and documents. To provide quantitative assessment of the performance of our method, that goes beyond visual inspection, we apply it to problems where some underlying structures are known in advance. The known structures are only used for performance measurement and not during learning.

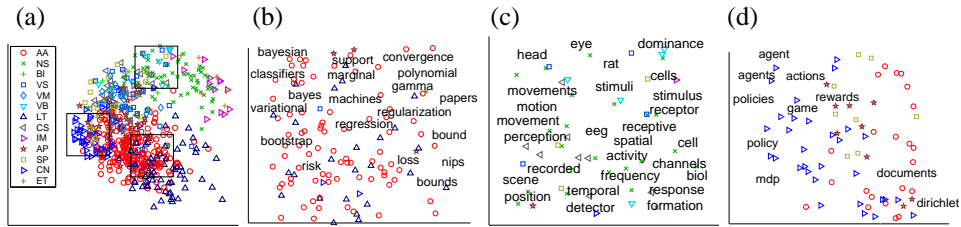

Figure 2: CODE Embedding of 2483 documents and 2000 words from the NIPS database (the 2000 most frequent words, excluding the first 100, were used). The left panel shows document embeddings for NIPS 15-17, with colors to indicate the document topic. Other panels show embedded words and documents for the areas specified by rectangles. Figure (b) shows the border region between algorithms and architecture (AA) and learning theory (LT) (bottom rectangle in (a)). Figure (c) shows the border region between neuroscience (NS) and biological vision (VB) (upper rectangle in (a)). Figure (d) shows mainly control and navigation (CN) documents (left rectangle in (a)).

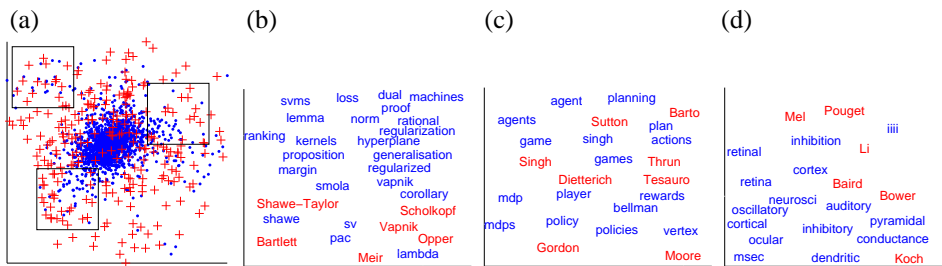

Figure 3: CODE Embedding of 2000 words and 250 authors from the NIPS database (the 250 authors with highest word counts were chosen; words were selected as in Figure 2). Left panel shows embeddings for authors (red crosses) and words (blue dots). Other panels show embedded authors (only first 100 shown) and words for the areas specified by rectangles. They can be seen to correspond to learning theory, control and neuroscience (from left to right).

## 5.1 NIPS Database

Embedding algorithms may be used to study the structure of document databases. Here we used the NIPS 0-12 database supplied by Roweis [2], and augmented it with data from NIPS volumes 13-17 [3]. The last three volumes also contain an indicator of the document's topic (AA for algorithms and architecture, LT for learning theory, NS for neuroscience etc.).

We first used CODE to embed documents and words into $\mathbb{R}^2$. The results are shown in Figure 2. It can be seen that documents with similar topics are mapped next to each other (e.g. AA near LT and NS near Biological Vision). Furthermore, words characterize the topics of their neighboring documents.

Next, we used the data to generate an authors-words matrix (as in the Roweis database). We could now embed authors and words into $\mathbb{R}^2$, by using CODE to model $p(word|author)$. The results are shown in Figure 3. It can be seen that authors are indeed mapped next to terms relevant to their work, and that authors dealing with similar domains are also mapped together. This illustrates how co-occurrence of words and authors may be used to induce a metric on authors alone.

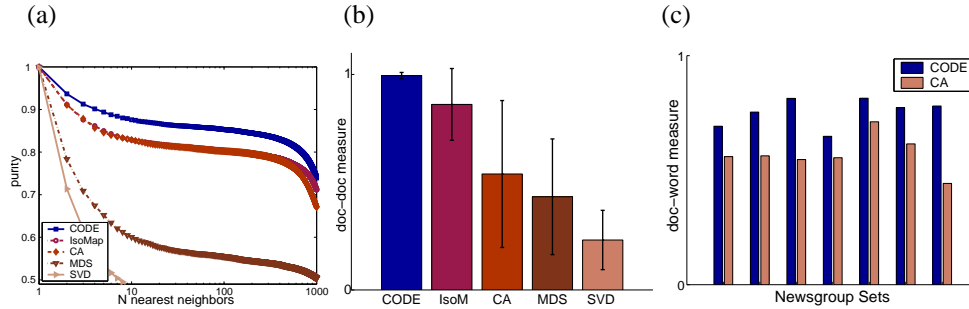

Figure 4: (a) Document purity measure for the embedding of newsgroups crypt, electronics and med, as a function of neighborhood size. (b) The $doc - doc$ measure averaged over 7 newsgroup sets. For each set, the maximum performance was normalized to one. Embedding dimension is 2. Sets are atheism, graphics, crypt; ms-windows, graphics; ibm.pc.hw, ms-windows; crypt, electronics; crypt, electronics, med; crypt, electronics, med, space; politics.mideast, politics.misc. (c) The $word - doc$ measure for CODE and CA algorithms, for 7 newsgroup sets. Embedding dimension is 2.

## 5.2 Information Retrieval

To obtain a more quantitative estimate of performance, we applied CODE to the 20 newsgroups corpus, preprocessed as described in [3]. This corpus consists of 20 groups, each with 1000 documents. We first removed the 100 most frequent words, and then selected the next $k$ most frequent words for different values of $k$ (see below). The resulting words and documents were embedded with CODE, Correspondence Analysis (CA), SVD, IsoMap and classical MDS [4]. CODE was used to model the distribution of words given documents $p(w|d)$. All methods were tested under several normalization schemes, including document sum normalization and TFIDF. Results were consistent over all normalization schemes.

An embedding of words and documents is expected to map documents with similar semantics together, *and* to map words close to documents which are related to the meaning of the word. We next test how our embeddings performs with respect to these requirements. To represent the *meaning* of a document we use its corresponding newsgroup. Note that this information is used only for evaluation and not in constructing the embedding itself.

To measure how well similar documents are mapped together we define a purity measure, which we denote $doc - doc$. For each embedded document, we measure the fraction of its neighbors that are from the same newsgroup. This is repeated for all neighborhood sizes, and averaged over all sizes and documents.

To measure how documents are related to their neighboring words, we use a measured denoted by $word - doc$. For each document $d$ we look at its $n$ nearest words and calculate their probability under the document's newsgroup, normalized by their prior. This is repeated for neighborhood sizes smaller than 100 and averaged over documents . The $word - doc$ measure was only compared with CA, since this is the only method that provides joint embeddings.

Figure 4 compares the performance of CODE with that of the other methods with respect to the $doc - doc$ and $word - doc$ measures. CODE can be seen to outperform all other methods on both measures.

# 6 Discussion

We presented a method for embedding objects of different types into the same low dimension Euclidean space. This embedding can be used to reveal low dimensional structures when distance measures between objects are unknown. Furthermore, the embedding induces a meaningful metric also between objects of the same type, which could be used, for example, to embed images based on accompanying text, and derive the semantic distance between images.

Co-occurrence embedding should not be restricted to pairs of variables, but can be extended to multivariate joint distributions, when these are available. It can also be augmented to use distances between same-type objects when these are known.

An important question in embedding objects is whether the embedding is unique. In other words, can there be two non isometric embeddings which are obtained at the optimum of the problem. This question is related to the rigidity and uniqueness of embeddings on graphs, specifically complete bipartite graphs in our case. A theorem of Bolker and Roth [1] asserts that for such graphs with at least 5 vertices on each side, embeddings are rigid, i.e. they cannot be continuously transformed. This suggests that the CODE embeddings for $|X|, |Y| \geq 5$ are unique (at least locally) for $d \leq 3$.

We focused here on geometric models for *conditional* distributions. While in some cases, such a modeling choice is more natural in others joint models may be more appropriate. In this context it will be interesting to consider models of the form $p(x,y) \propto p(x)p(y)e^{-d_{x,y}^2}$ where $p(x), p(y)$ are the marginals of $p(x,y)$. Maximum likelihood in these models is a non-trivial constrained optimization problem, and may be approached using the semidefinite representation outlined here.

## Footnotes

[1]We have studied several other models of the joint rather than the conditional distribution. These differ by the way the marginals are modeled and will be described elsewhere

[2] See `http://www.cs.toronto.edu/~roweis/data.html`

[3] Data available at `http://robotics.stanford.edu/~gal/`

[4]CA embedding followed the standard procedure described in [8]. IsoMap implementation was provided by the IsoMap authors [13]. We tested both an SVD over the count matrix and SVD over log of the count plus one, only the latter is described here because it was better than the former. For MDS, the distances between objects were calculated as the dot product between their count vectors (we also tested Euclidean distances)

# References

[1] E.D. Bolker. and B. Roth. When is a bipartite graph a rigid framework? *Pacific J. Math.*, 90:27–44, 1980.

[2] S. Boyd and L. Vandenberghe. *Convex Optimization*. Cambridge Univ. Press, 2004.

[3] G. Chechik and N. Tishby. Extracting relevant structures with side information. In S. Becker, S. Thrun, and K. Obermayer, editors, *NIPS 15*, 2002.

[4] T. Cox and M. Cox. *Multidimensional Scaling*. Chapman and Hall, London, 1984.

[5] M. Fazel, H. Hindi, and S. P. Boyd. A rank minimization heuristic with application to minimum order system approximation. In *Proc. of the American Control Conference*, 2001.

[6] R.A. Fisher. The percision of discriminant functions. *Ann. Eugen. Lond.*, 10:422–429, 1940.

[7] A. Globerson and N. Tishby. Sufficient dimensionality reduction. *Journal of Machine Learning Research*, 3:1307–1331, 2003.

[8] M.J. Greenacre. *Theory and applications of correspondence analysis*. Academic Press, 1984.

[9] G. Hinton and S.T. Roweis. Stochastic neighbor embedding. In *NIPS 15*, 2002.

[10] H. Hotelling. The most predictable criterion. *Journal of Educational Psych.*, 26:139–142, 1935.

[11] T. Iwata, K. Saito, N. Ueda, S. Stromsten, T. Griffiths, and J. Tenenbaum. Parametric embedding for class visualization. In *NIPS 18*, 2004.

[12] S. T. Roweis and L. K. Saul. Nonlinear dimensionality reduction by locally linear embedding. *Science*, 290:2323–2326, 2000.

[13] J.B. Tenenbaum, V. de Silva, and J. C. Langford. A global geometric framework for nonlinear dimensionality reduction. *Science*, 290:2319–2323, 2000.

[14] K. Q. Weinberger and L. K. Saul. Unsupervised learning of image manifolds by semidefinite programming. In *CVPR*, 2004.
